# On $U$-processes and clustering performance

**Stéphan Clémençon**[*]
LTCI UMR Telecom ParisTech/CNRS No. 5141
Institut Telecom, Paris, 75634 Cedex 13, France
stephan.clemencon@telecom-paristech.fr

## Abstract

Many clustering techniques aim at optimizing empirical criteria that are of the form of a $U$-statistic of degree two. Given a measure of dissimilarity between pairs of observations, the goal is to minimize the *within cluster* point scatter over a class of partitions of the feature space. It is the purpose of this paper to define a general statistical framework, relying on the theory of $U$-processes, for studying the performance of such clustering methods. In this setup, under adequate assumptions on the complexity of the subsets forming the partition candidates, the *excess of clustering risk* is proved to be of the order $O_{\mathbb{P}}(1/\sqrt{n})$. Based on recent results related to the tail behavior of degenerate $U$-processes, it is also shown how to establish tighter rate bounds. Model selection issues, related to the number of clusters forming the data partition in particular, are also considered.

## 1 Introduction

In *cluster analysis*, the objective is to segment a dataset into subgroups, such that data points in the same subgroup are more similar to each other (in a sense that will be specified) than to those in other subgroups. Given the wide range of applications of the clustering paradigm, numerous data segmentation procedures have been introduced in the machine-learning literature (see Chapter 14 in [HTF09] and Chapter 8 in [CFZ09] for recent overviews of "off-the-shelf" clustering techniques). Whereas the design of clustering algorithms is still receiving much attention in machine-learning (see [WT10] and the references therein for instance), the statistical study of their performance, with the notable exception of the celebrated $K$-means approach, see [Har78, Pol81, Pol82, BD04] and more recently [BDL08] in the functional data analysis setting, may appear to be not sufficiently well-documented in contrast, as pointed out in [vLBD05, BvL09]. Indeed, in the $K$-means situation, the specific form of the criterion (and of its expectation, the *clustering risk*), as well as that of the cells defining the clusters and forming a partition of the feature space (*Voronoi cells*), permits to use, in a straightforward manner, results of the theory of empirical processes in order to control the performance of empirical clustering risk minimizers. Unfortunately, this *center-based* approach does not carry over into more general situations, where the dissimilarity measure is not a square hilbertian norm anymore, unless one loses the possibility to interpret the clustering criterion as a function of pairwise dissimilarities between the observations (*cf $K$-medians*).

It is the goal of this paper to establish a general statistical framework for investigating clustering performance. The present analysis is based on the observation that many statistical criteria for measuring clustering accuracy are (symmetric) $U$-statistics (of degree two), functions of a matrix of dissimilarities between pairs of data points. Such statistics have recently received a good deal of attention in the machine-learning literature, insofar as empirical performance measures of predictive rules in problems such as *statistical ranking* (when viewed as pairwise classification), see [CLV08], or *learning on graphs* ([BB06]), are precisely functionals of this type, generalizing sample mean statistics. By means of uniform deviation results for $U$-processes, the *Empirical Risk Minimization*

---

[*]http://www.tsi.enst.fr/~clemenco/.

paradigm (ERM) can be extended to situations where natural estimates of the risk are $U$-statistics. In this way, we establish here a rate bound of order $O_{\mathbb{P}}(1/\sqrt{n})$ for the excess of clustering risk of empirical minimizers under adequate complexity assumptions on the cells forming the partition candidates (the bias term is neglected in the present analysis). A linearization technique, combined with sharper tail results in the case of degenerate $U$-processes is also used in order to show that tighter rate bounds can be obtained. Finally, it is shown how to use the upper bounds established in this analysis in order to deal with the problem of automatic model selection, that of selecting the number of clusters in particular, through complexity penalization.

The paper is structured as follows. In section 2, the notations are set out, a formal description of cluster analysis, from the "pairwise dissimilarity" perspective, is given and the main theoretical concepts involved in the present analysis are briefly recalled. In section 3, an upper bound for the performance of empirical minimization of the clustering risk is established in the context of general dissimilarity measures. Section 4 shows how to refine the rate bound previously obtained by means of a recent inequality for degenerate $U$-processes, while section 5 deals with automatic selection of the optimal number of clusters. Technical proofs are deferred to the Appendix section.

## 2   Theoretical background

In this section, after a brief description of the probabilistic framework of the study, the general formulation of the clustering objective, based on the notion of dissimilarity between pairs of observations, is recalled and the connection of the problem of investigating clustering performance with the theory of $U$-statistics and $U$-processes is highlighted. Concepts pertaining to this theory and involved in the subsequent analysis are next recalled.

### 2.1   Probabilistic setup and first notations

Here and throughout, $(X_1, \ldots, X_n)$ denotes a sample of i.i.d. random vectors, valued in a high-dimensional feature space $\mathcal{X}$, typically a subset of the euclidian space $\mathbb{R}^d$ with $d >> 1$, with common probability distribution $\mu(dx)$. With no loss of generality, we assume that the feature space $\mathcal{X}$ coincides with the support of the distribution $\mu(dx)$. The indicator function of any event $\mathcal{E}$ will be denoted by $\mathbb{I}\{\mathcal{E}\}$, the usual $l_p$ norm on $\mathbb{R}^d$ by $||x||_p = (\sum_{i=1}^d |x_i|^p)^{1/p}$ when $1 \le p < \infty$ and by $||x||_\infty = \max_{1 \le i \le d} |x_i|$ in the case $p = \infty$, with $x = (x_1, \ldots, x_d) \in \mathbb{R}^d$. When well-defined, the expectation and the variance of a r.v. $Z$ are denoted by $\mathbb{E}[Z]$ and $\mathrm{Var}(Z)$ respectively. Finally, we denote by $x_+ = \max(0, x)$ the positive part of any real number $x$.

### 2.2   Cluster analysis

The goal of clustering techniques is to partition the data $(X_1, \ldots, X_n)$ into a given finite number of groups, $K << n$ say, so that the observations lying in a same group are more similar to each other than to those in other groups. When equipped with a (borelian) measure of dissimilarity $D : \mathcal{X}^2 \to \mathbb{R}_+^*$, the clustering task can be rigorously cast as the problem of minimizing the criterion

$$\widehat{W}_n(\mathcal{P}) = \frac{2}{n(n-1)} \sum_{k=1}^K \sum_{1 \le i < j \le n} D(X_i, X_j) \cdot \mathbb{I}\{(X_i, X_j) \in \mathcal{C}_k^2\}, \tag{1}$$

over all possible partitions $\mathcal{P} = \{\mathcal{C}_k : 1 \le k \le K\}$ of the feature space $\mathcal{X}$. The quantity (1) is generally called the *intra-cluster similarity* or the *within cluster point scatter*. The function $D$ aiming at measuring dissimilarity between pairs of observations, we suppose that it fulfills the following properties:

- (SYMMETRY) For all $(x, x') \in \mathcal{X}^2$, $D(x, x') = D(x', x)$
- (SEPARATION) For all $(x, x') \in \mathcal{X}^2$: $D(x, x') = 0, \Leftrightarrow x = x'$

Typical choices for the dissimilarity measure are of the form $D(x, x') = \Phi(||x - x'||_p)$, where $p \ge 1$ and $\Phi : \mathbb{R}_+ \to \mathbb{R}_+$ is a nondecreasing function such that $\Phi(0) = 0$ and $\Phi(t) > 0$ for all $t > 0$. This includes the so-termed "standard $K$-means" setup, where the dissimilarity measure coincides with

the square euclidian norm (in this case, $p = 2$ and $\Phi(t) = t^2$ for $t \geq 0$). Notice that the expectation of the r.v. (1) is equal to the following quantity:

$$W(\mathcal{P}) = \sum_{k=1}^{K} \mathbb{E} \left[ D(X, X') \cdot \mathbb{I}\{(X, X') \in \mathcal{C}_k^2\} \right], \tag{2}$$

where $(X, X')$ denotes a pair of independent r.v.'s drawn from $\mu(dx)$. It will be referred to as the *clustering risk* of the partition $\mathcal{P}$, while its statistical counterpart (1) will be called the *empirical clustering risk*. *Optimal partitions* of the feature space $\mathcal{X}$ are defined as those that minimize $W(\mathcal{P})$.

**Remark 1** (MAXIMIZATION FORMULATION) It is well-known that minimizing the empirical clustering risk (1) is equivalent to maximizing the *between-cluster* scatter point, which is given by $1/(n(n-1)) \cdot \sum_{k \neq l} \sum_{i, j} D(X_i, X_j) \cdot \mathbb{I}\{(X_i, X_j) \in \mathcal{C}_k \times \mathcal{C}_l\}$, the sum of these two statistics being independent from the partition $\mathcal{P} = \{\mathcal{C}_k : 1 \leq k \leq K\}$ considered.

Suppose we are given a (hopefully sufficiently rich) class $\Pi$ of partitions of the feature space $\mathcal{X}$. Here we consider minimizers of the empirical risk $\widehat{W}_n$ over $\Pi$, *i.e.* partitions $\widehat{\mathcal{P}}_n^*$ in $\Pi$ such that

$$\widehat{W}_n \left( \widehat{\mathcal{P}}_n^* \right) = \min_{\mathcal{P} \in \Pi} \widehat{W}_n \left( \mathcal{P} \right). \tag{3}$$

The design of practical algorithms for computing (approximately) empirical clustering risk minimizers is beyond the scope of this paper (refer to [HTF09] for an overview of "off-the-shelf" clustering methods). Here, focus is on the performance of such empirically defined rules.

## 2.3 $U$-statistics and $U$-processes

The subsequent analysis crucially relies on the fact that the quantity (1) that one seeks to optimize is a $U$-statistic. For clarity's sake, we recall the definition of this class of statistics, generalizing sample means.

**Definition 1** ($U$-STATISTIC OF DEGREE TWO.) *Let $X_1$, ..., $X_n$ be independent copies of a random vector $X$ drawn from a probability distribution $\mu(dx)$ on the space $\mathcal{X}$ and $\mathcal{K} : \mathcal{X}^2 \to \mathbb{R}$ be a symmetric function such that $\mathcal{K}(X_1, X_2)$ is square integrable. By definition, the functional*

$$U_n = \frac{2}{n(n-1)} \sum_{1 \leq i < j \leq n} \mathcal{K}(X_i, X_j). \tag{4}$$

*is a (symmetric) $U$-statistic of degree two, with kernel $\mathcal{K}$. It is said to be degenerate when $\mathcal{K}^{(1)}(x) \stackrel{def}{=} \mathbb{E}[\mathcal{K}(x, X)] = 0$ with probability one for all $x \in \mathcal{X}$, non degenerate otherwise.*

The statistic (4) is a natural (unbiased) estimate of the quantity $\theta = \int \int \mathcal{K}(x, x') \mu(dx) \mu(dx')$. The class of $U$-statistics is very large and include most dispersion measures, including the variance or the Gini mean difference (with $\mathcal{K}(x, x') = (x - x')^2$ and $\mathcal{K}(x, x') = |x - x'|$ respectively, $(x, x') \in \mathbb{R}^2$), as well as the celebrated Wilcoxon location test statistic (with $\mathcal{K}(x, x') = \mathbb{I}\{x + x' > 0\}$ for $(x, x') \in \mathbb{R}^2$ in this case). Although the dependence structure induced by the summation over all pairs of observations makes its study more difficult than that of basic sample means, this estimator has nice properties. It is well-known folklore in mathematical statistics that it is the most *efficient* estimator among all unbiased estimators of the parameter $\theta$ (*i.e.* that with minimum variance), see [vdV98]. Precisely, when non degenerate, it is asymptotically normal with limiting variance $4 \cdot \mathrm{Var}(\mathcal{K}^{(1)}(X))$ (refer to Chapter 5 in [Ser80] for an account of asymptotic analysis of $U$-statistics). As shall be seen in section 4, the reduced variance property of $U$-statistics is crucial, when it comes to establish tight rate bounds.

Going back to the $U$-statistic of degree two (1) estimating (2), observe that its symmetric kernel is:

$$\forall (x, x') \in \mathcal{X}^2, \ \mathcal{K}_{\mathcal{P}}(x, x') = \sum_{k=1}^{K} D(x, x') \cdot \mathbb{I}\{(x, x') \in \mathcal{C}_k^2\}. \tag{5}$$

Assuming that $\mathbb{E}[D^2(X_1, X_2) \cdot \mathbb{I}\{(X_1, X_2) \in \mathcal{C}_k^2\}] < \infty$ for all $k \in \{1, ..., K\}$ and placing ourselves in the situation where $K \geq 1$ is less than $\mathcal{X}$'s cardinality, the $U$-statistic (1) is always non

degenerate, except in the (sole) case where $\mathcal{X}$ is made of $K$ elements exactly and all $\mathcal{P}$'s cells are singletons. Indeed, for all $x \in \mathcal{X}$, denoting by $k(x)$ the index of $\{1, \ldots, K\}$ such that $x \in \mathcal{C}_{k(x)}$, we have:

$$\mathcal{K}_{\mathcal{P}}^{(1)}(x) \stackrel{def}{=} \mathbb{E}[\mathcal{K}_{\mathcal{P}}(x, X)] = \int_{x' \in \mathcal{C}_{k(x)}} D(x, x') \mu(dx'). \qquad (6)$$

As $\mu$'s support coincides with $\mathcal{X}$ and the separation property is fulfilled by $D$, the quantity above is zero iff $C_{k(x)} = \{x\}$. In the non degenerate case, notice finally that the asymptotic variance of $\sqrt{n}\{\widehat{W}_n(\mathcal{P}) - W(\mathcal{P})\}$ is equal to $4 \cdot \mathrm{Var}(D(X, \mathcal{C}_{k(X)}))$, where we set $D(x, C) = \int_{x' \in X} D(x, x') \mu(dx')$ for all $x \in \mathcal{X}$ and any measurable set $C \subset \mathcal{X}$.

By definition, a $U$-process is a collection of $U$-statistics, one may refer to [dlPG99] for an account of the theory of $U$-processes. Echoing the role played by the theory of empirical processes in the study of the ERM principle in binary classification, the control of the fluctuations of the $U$-process

$$\left\{ \widehat{W}_n(\mathcal{P}) - W(\mathcal{P}) : \ \mathcal{P} \in \Pi \right\}$$

indexed by a set $\Pi$ of partition candidates will naturally lie at the heart of the present analysis. As shall be seen below, this can be achieved mainly by the means of the Hoeffding representations of $U$-statistics, see [Hoe48].

## 3  A bound for the excess of clustering risk

Here we establish an upper bound for the performance of an empirical minimizer of the clustering risk over a class $\Pi_K$ of partitions of $\mathcal{X}$ with $K \geq 1$ cells, $K$ being fixed here and supposed to be smaller than $\mathcal{X}$'s cardinality. We denote by $W_K^*$ the clustering risk minimum over all partitions of $\mathcal{X}$ with $K$ cells. The following global suprema of empirical Rademacher averages, characterizing the complexity of the cells forming the partition candidates, shall be involved in the subsequent rate analysis: $\forall n \geq 2$,

$$\mathcal{A}_{K,n} = \sup_{\mathcal{C} \in \mathcal{P}, \ \mathcal{P} \in \Pi_K} \frac{1}{\lfloor n/2 \rfloor} \left| \sum_{i=1}^{\lfloor n/2 \rfloor} \epsilon_i D(X_i, X_{i+\lfloor n/2 \rfloor}) \cdot \mathbb{I}\{(X_i, X_{i+\lfloor n/2 \rfloor}) \in \mathcal{C}^2\} \right|, \qquad (7)$$

where $\epsilon = (\epsilon_i)_{i \geq 1}$ is a Rademacher chaos, independent from the $X_i$'s, see [Kol06].

The following theorem reveals that the clustering performance of the empirical minimizer (3) is of the order $O_{\mathbb{P}}(1/\sqrt{n})$, when neglecting the bias term (depending on the richness of $\Pi_K$ solely).

**Theorem 1** *Consider a class $\Pi_K$ of partitions with $K \geq 1$ cells and suppose that:*

- *there exists $B < \infty$ such that for all $\mathcal{P}$ in $\Pi_K$, any $\mathcal{C}$ in $\mathcal{P}$, $\sup_{(x,x') \in \mathcal{C}^2} D(x, x') \leq B$,*

- *the expectation of the Rademacher average $\mathcal{A}_{K,n}$ is of the order $O(n^{-1/2})$.*

*Let $\delta > 0$. For any empirical clustering risk minimizer $\widehat{\mathcal{P}}_n^*$, we have with probability at least $1 - \delta$:*

$$\forall n \geq 2, \ W(\widehat{\mathcal{P}}_n^*) - W_K^* \ \leq \ 4K\mathbb{E}[\mathcal{A}_{K,n}] + 2BK\sqrt{\frac{2\log(1/\delta)}{n}} + \left( \inf_{\mathcal{P} \in \Pi_K} W(\mathcal{P}) - W_K^* \right)$$

$$\leq \ c(B, \delta) \cdot \frac{K}{\sqrt{n}} + \left( \inf_{\mathcal{P} \in \Pi_K} W(\mathcal{P}) - W_K^* \right), \qquad (8)$$

*for some constant $c(B, \delta) < \infty$, independent from $n$ and $K$.*

The key for proving (8) is to express the $U$-statistic $W_n(\mathcal{P})$ in terms of sums of i.i.d. r.v.'s, as that involved in the Rademacher average (7):

$$W_n(\mathcal{P}) = \frac{1}{n!} \sum_{\sigma \in \mathfrak{S}_n} \frac{1}{\lfloor n/2 \rfloor} \sum_{i=1}^{\lfloor n/2 \rfloor} \mathcal{K}_{\mathcal{P}}(X_i, X_{i+\lfloor n/2 \rfloor}), \qquad (9)$$

where the average is taken over $\mathfrak{S}_n$, the symmetric group of order $n$. The main point lies in the fact that standard techniques in empirical process theory can be then used to control $W_n(\mathcal{P}) - W(\mathcal{P})$ uniformly over $\Pi_K$ under adequate hypotheses, see the proof in the Appendix for technical details. We underline that, naturally, the complexity assumption is also a crucial ingredient of the result stated above, and more generally to clustering consistency results, see Example 1 in [BvL09]. We also point out that the ERM approach is by no means the sole method to obtain error bounds in the clustering context. Just like in binary classification (see [KN02]), one may use a notion of *stability* of a clustering algorithm to establish such results, see [vL09, ST09] and the references therein. Refer to [vLBD06, vLBD08] for error bounds proved through the stability approach. Before showing how the bound for the excess of risk stated above can be improved, a few remarks are in order.

**Remark 2** (ON THE COMPLEXITY ASSUMPTION.) We point out that standard entropy metric arguments can be used in order to bound the expected value of the Rademacher average $\mathcal{A}_n$, see [BBL05] for instance. In particular, if the set of functions $\mathcal{F}_{\Pi_K} = \{(x, x') \in \mathcal{X}^2 \mapsto D(x, x') \cdot \mathbb{I}\{(x, x') \in \mathcal{C}^2\} : \mathcal{C} \in \mathcal{P}, \ \mathcal{P} \in \Pi_K\}$ is a VC major class with finite VC dimension V (see [Dud99]), then $\mathbb{E}[\mathcal{A}_{K,n}] \leq c\sqrt{V/n}$ for some universal constant $c < \infty$. This covers a wide variety of situations, including the case where $D(x, x') = ||x - x'||_p^\beta$ and the class of sets $\{\mathcal{C} : \mathcal{C} \in \mathcal{P}, \ \mathcal{P} \in \Pi_K\}$ is of finite VC dimension.

**Remark 3** ($K$-MEANS.) In the standard $K$-means approach, the dissimilarity measure is $D(x, x') = ||x - x'||_2^2$ and partition candidates are indexed by a collection $c$ of distinct "centers" $c_1, \ldots, c_K$ in $\mathcal{X}$: $\mathcal{P}_c = \{C_1, \ldots, C_K\}$ with $C_k = \{x \in \mathcal{X} : ||x - c_k||_2 = \min_{1 \leq l \leq K} ||x - c_l||_2\}$ for $1 \leq k \leq K$ (with adequate distance-tie breaking). One may easily check that for this specific collection of partitions $\Pi_K$ and this choice for the dissimilarity measure, the class $\mathcal{F}_{\Pi_K}$ is a VC major class with finite VC dimension, see section 19.1 in [DGL96] for instance. Additionally, it should be noticed than in most practical clustering procedures, center candidates are picked in a data-driven fashion, being taken as the averages of the observations lying in each cluster/cell. In this respect, the $M$-estimation problem formulated here can be considered to a certain extent as closer to what is actually achieved by $K$-means clustering techniques in practice, than the usual formulation of the $K$-means problem (as an optimization problem over $c = (c_1, \ldots, c_K)$ namely).

**Remark 4** (WEIGHTED CLUSTERING CRITERIA.) Notice that, in practice, the measure $D$ involved in (1) may depend on the data. For scaling purpose, one could assign data-dependent weights $\omega = (\omega_i)_{1 \leq i \leq d}$ in a coordinatewise manner, leading to $\widehat{D}(x, x') = \sum_{i=1}^{d}(x_i - x_i')^2/\widehat{\sigma}_i^2$ for instance, where $\widehat{\sigma}_i^2$ denotes the sample variance related to the $i$-th coordinate. Although the criterion reflecting the performance is not a $U$-statistic anymore, the theory we develop here can be straightforwardly used for investigating clustering accuracy in such a case. Indeed, it is easy to control the difference between the latter and the $U$-statistic (1) with $D(x, x') = \sum_{i=1}^{d}(x_i - x_i')^2/\sigma_i^2$, the $\sigma_i^2$'s denoting the theoretical variances of $\mu$'s marginals, under adequate moment assumptions.

## 4 Tighter bounds for empirical clustering risk minimizers

We now show that one may refine the rate bound established above, by considering another representation of the $U$-statistic (1), its *Hoeffding's decomposition* (see [Ser80]): for all partition $\mathcal{P}$,

$$W_n(\mathcal{P}) - W(\mathcal{P}) = 2L_n(\mathcal{P}) + M_n(\mathcal{P}), \tag{10}$$

$L_n(\mathcal{P}) = (1/n)\sum_{i=1}^{n}\sum_{\mathcal{C} \in \mathcal{P}} \mathcal{H}_{\mathcal{C}}^{(1)}(X_i)$ being a simple average of i.i.d r.v.'s with, for $(x, x') \in \mathcal{X}^2$,

$$\mathcal{H}_{\mathcal{C}}(x, x') = D(x, x') \cdot \mathbb{I}\{(x, x') \in \mathcal{C}^2\} \text{ and } \mathcal{H}_{\mathcal{C}}^{(1)}(x) = D(x, \mathcal{C}) \cdot \mathbb{I}\{x \in \mathcal{C}\} - D(\mathcal{C}, \mathcal{C}),$$

where $D(\mathcal{C}, \mathcal{C}) = \int_{x \in \mathcal{C}} D(x, \mathcal{C})\mu(dx)$ and $\mathbb{E}[\mathcal{H}_{\mathcal{C}}(x, X)] = D(x, \mathcal{C}) \cdot \mathbb{I}\{x \in \mathcal{C}\}$, and $M_n(\mathcal{P})$ being a degenerate $U$-statistic based on the $X_i$'s with kernel given by: $\sum_{\mathcal{C} \in \mathcal{P}} \mathcal{H}_{\mathcal{C}}^{(2)}(x, x')$, where

$$\mathcal{H}_{\mathcal{C}}^{(2)}(x, x') = \mathcal{H}_{\mathcal{C}}(x, x') - \mathcal{H}_{\mathcal{C}}^{(1)}(x) - \mathcal{H}_{\mathcal{C}}^{(1)}(x') - D(\mathcal{C}, \mathcal{C}),$$

for all $(x, x') \in \mathcal{X}^2$. The leading term in (10) is the (centered) sample mean $2L_n(\mathcal{P})$, of the order $O_{\mathbb{P}}(\sqrt{1/n})$, while the second term is of the order $O_{\mathbb{P}}(1/n)$. Hence, provided this holds true

uniformly over $\mathcal{P}$, the main contribution to the rate bound should arise from the quantity

$$\sup_{\mathcal{P}\in\Pi_K} |2L_n(\mathcal{P})| \le 2K \sup_{\mathcal{C}\in\mathcal{P},\ \mathcal{P}\in\Pi_K} |(1/n)\sum_{i=1}^n \mathcal{H}_{\mathcal{C}}^{(1)}(X_i) - D(\mathcal{C},\mathcal{C})|,$$

which thus leads to consider the following suprema of empirical Rademacher averages:

$$\mathcal{R}_{K,n} = \sup_{\mathcal{C}\in\mathcal{P},\ \mathcal{P}\in\Pi_K} \frac{1}{n}\left|\sum_{i=1}^n \epsilon_i D(X_i,\mathcal{C})\cdot\mathbb{I}\{X_i\in\mathcal{C}\}\right|. \tag{11}$$

This supremum clearly has smaller mean and variance than (7). We also introduce the quantities:

$$Z_\epsilon = \sup_{\mathcal{C}\in\mathcal{P},\ \mathcal{P}\in\Pi_K} \left|\sum_{i,j}\epsilon_i\epsilon_j\mathcal{H}_{\mathcal{C}}^{(2)}(X_i,X_j)\right|,\ U_\epsilon = \sup_{\mathcal{C}\in\mathcal{P},\ \mathcal{P}\in\Pi_K}\sup_{\alpha:\ \sum_j\alpha_j^2}\sum_{i,j}\epsilon_i\alpha_j\mathcal{H}_{\mathcal{C}}^{(2)}(X_i,X_j),$$

$$M = \sup_{\mathcal{C}\in\mathcal{P},\ \mathcal{P}\in\Pi_K}\sup_{1\le j\le n}\left|\sum_i\epsilon_i\mathcal{H}_{\mathcal{C}}^{(2)}(X_i,X_j)\right|.$$

**Theorem 2** *Consider a class $\Pi_K$ of partitions with $K$ cells and suppose that:*

- *there exists $B < \infty$ such that $\sup_{(x,x')\in\mathcal{C}^2} D(x,x') \le B$ for all $\mathcal{P}\in\Pi_K$, $\mathcal{C}\in\mathcal{P}$.*

*Let $\delta > 0$. For any empirical clustering risk minimizer $\widehat{\mathcal{P}}_n^*$, with probability at least $1 - \delta$: $\forall n \ge 2$,*

$$W(\widehat{\mathcal{P}}_n^*) - W_K^* \le 4K\mathbb{E}[\mathcal{R}_{K,n}] + 2BK\sqrt{\frac{\log(2/\delta)}{n}} + K\kappa(n,\delta) + \left(\inf_{\mathcal{P}\in\Pi_K} W(\mathcal{P}) - W_K^*\right), \tag{12}$$

*where we set for some universal constant $C < \infty$, independent from $n$, $N$ and $K$:*

$$\kappa(n,\delta) = C\left(\mathbb{E}[Z_\epsilon] + \sqrt{\log(1/\delta)}\mathbb{E}[U_\epsilon] + (n+\mathbb{E}[M])/\log(1/\delta)\right)/n^2. \tag{13}$$

The result above relies on the moment inequality for degenerate $U$-processes proved in [CLV08].

**Remark 5** (LOCALIZATION.) The same argument can be used to decompose $\Lambda_n(\mathcal{P}) - \Lambda(\mathcal{P})$, where $\Lambda_n(\mathcal{P}) = \widehat{W}_n(\mathcal{P}) - W_K^*$ is an estimate of the excess of risk $\Lambda(\mathcal{P}) = W(\mathcal{P}) - W_K^*$, and, by means of concentration inequalities, to obtain next a sharp upper bound that involves the modulus of continuity of the variance of the Rademacher average indexed by the convex hull of the set of functions $\{\sum_{\mathcal{C}\in\mathcal{P}} D(x,\mathcal{C})\cdot\mathbb{I}\{x\in\mathcal{C}\} - \sum_{\mathcal{C}^*\in\mathcal{P}^*} D(x,\mathcal{C}^*)\cdot\{x\in\mathcal{C}^*\} : \mathcal{P}\in\Pi_K\}$, following in the footsteps or recent advances in binary classification, see [Kol06] and subsection 5.3 in [BBL05]. Owing to space limitations, this will be dealt with in a forthcoming article.

## 5 Model selection - choosing the number of clusters

A crucial issue in data segmentation is to determine the number $K$ of cells that exhibits the most the clustering phenomenon in the data. A variety of automatic procedures for choosing a good value for $K$ have been proposed in the literature, based on data splitting, resampling or sampling techniques ([PFvN89, TWH01, ST08]). Here we consider a complexity regularization method that avoids to have recourse to such techniques and uses a data-dependent penalty term based on the analysis carried out above.

Suppose that we have a sequence $\Pi_1$, $\Pi_2$, ... of collections of partitions of the feature space $\mathcal{X}$ such that, for all $K \ge 1$, the elements of $\Pi_K$ are made of $K$ cells and fulfill the assumptions of Theorem 1. In order to avoid overfitting, consider the (data-driven) complexity penalty given by

$$pen(n,K) = 3K\mathbb{E}_\epsilon[\mathcal{A}_{K,n}] + \frac{27BK\log K}{n} + \sqrt{(2B\log K)/n} \tag{14}$$

and the minimizer $\widehat{\mathcal{P}}_{\widehat{K},n}$ of the penalized empirical clustering risk, with

$$\widehat{K} = \arg\min_{K\ge 1}\left\{\widehat{W}_n(\widehat{\mathcal{P}}_{K,n}) + pen(n,K)\right\}\ \text{and}\ \widehat{W}_n(\widehat{\mathcal{P}}_{K,n}) = \min_{\mathcal{P}\in\Pi_K}\widehat{W}_n(\mathcal{P}).$$

The next result shows that the partition thus selected nearly achieves the performance that would be obtained with the help of an oracle, revealing the value of the index $K$ that minimizes $\mathbb{E}[\widehat{\mathcal{P}}_{K,n}] - W^*$, with $W^* = \inf_{\mathcal{P}} W(\mathcal{P})$.

**Theorem 3** (AN ORACLE INEQUALITY) *Suppose that, for all $K \geq 1$, the assumptions of Theorem 1 are fulfilled. Then, we have:*

$$\mathbb{E}\left[W(\widehat{\mathcal{P}}_{\widehat{K},n})\right] - W^* \leq \min_{K \geq 1} \{W_K^* - W^* + pen(n,K)\} + \frac{\pi^2}{6}\left(2B\sqrt{\frac{2}{n}} + \frac{18B}{n}\right). \qquad (15)$$

Of course, the penalty could be slightly refined using the results of Section 4. Due to space limitations, such an analysis is not carried out here and is left to the reader.

## 6   Conclusion

Whereas, until now, the theoretical analysis of clustering performance was mainly limited to the $K$-means situation (but not only, *cf* [BvL09] for instance), this paper establishes bounds for the success of empirical clustering risk minimization in a general "pairwise dissimilarity" framework, relying on the theory of $U$-processes. The excess of risk of empirical minimizers of the clustering risk is proved to be of the order $O_{\mathbb{P}}(n^{-1/2})$ under mild assumptions on the complexity of the cells forming the partition candidates. It is also shown how to refine slightly this upper bound through a linearization technique and the use of recent inequalities for degenerate $U$-processes. Although the improvement displayed here can appear as not very significant at first glance, our approach suggests that much sharper data-dependent bounds could be established this way. To the best of our knowledge, the present analysis is the first to state results of this nature. As regards complexity regularization, while focus is here on the choice of the number of clusters, the argument used in this paper also paves the way for investigating more general model selection issues, including choices related to the geometry/complexity of the cells of the partition considered.

## Appendix - Technical proofs

**Proof of Theorem 1**

We may classically write:

$$\begin{aligned}
\widehat{W}(\widehat{\mathcal{P}}_n) - W_K^* &\leq 2\sup_{\mathcal{P} \in \Pi_K} |\widehat{W}_n(\mathcal{P}) - W(\mathcal{P})| + \left(\inf_{\mathcal{P} \in \Pi_K} W(\mathcal{P}) - W_K^*\right) \\
&\leq 2K \sup_{\mathcal{C} \in \mathcal{P}, \, \mathcal{P} \in \Pi_K} |U_n(\mathcal{C}) - u(\mathcal{C})| + \left(\inf_{\mathcal{P} \in \Pi_K} W(\mathcal{P}) - W_K^*\right), \qquad (16)
\end{aligned}$$

where $U_n(\mathcal{C})$ denotes the $U$-statistic with kernel $\mathcal{H}_{\mathcal{C}}(x, x') = D(x, x') \cdot \mathbb{I}\{(x, x') \in \mathcal{C}^2\}$ based on the sample $X_1, \ldots, X_n$ and $u(\mathcal{C})$ its expectation. Therefore, mimicking the argument of Corollary 3 in [CLV08], based on the so-termed *first Hoeffding's representation* of $U$-statistics (see Lemma A.1 in [CLV08]), we may straightforwardly derive the lemma below.

**Proposition 1** (UNIFORM DEVIATIONS) *Suppose that Theorem 1's assumptions are fulfilled. Let $\delta > 0$. With probability at least $1 - \delta$, we have: $\forall n \geq 2$,*

$$\sup_{\mathcal{C} \in \mathcal{P}, \, \mathcal{P} \in \Pi_K} |U_n(\mathcal{C}) - u(\mathcal{C})| \leq 2\mathbb{E}[\mathcal{A}_{K,n}] + B\sqrt{\frac{2\log(1/\delta)}{n}}. \qquad (17)$$

PROOF. The argument follows in the footsteps of Corollary 3's proof in [CLV08]. It is based on the so-termed *first Hoeffding's representation* of $U$-statistics (9), which provides an immediate control of the moment generating function of the supremum $\sup_{\mathcal{C}} |U_n(\mathcal{C}) - u(\mathcal{C})|$ by that of the norm of an empirical process, namely $\sup_{\mathcal{C}} |A_n(\mathcal{C}) - u(\mathcal{C})|$, where, for all $\mathcal{C} \in \mathcal{P}$ and $\mathcal{P} \in \Pi_K$:

$$A_n(\mathcal{C}) = \frac{1}{\lfloor n/2 \rfloor} \sum_{i=1}^{\lfloor n/2 \rfloor} D(X_i, X_{i+\lfloor n/2 \rfloor}) \cdot \mathbb{I}\{(X_i, X_{i+\lfloor n/2 \rfloor}) \in \mathcal{C}^2\}.$$

**Lemma 1** *(see Lemma A.1 in [CLV08]) Let $\Psi : \mathbb{R} \to \mathbb{R}$ be convex and nondecreasing. We have:*

$$\mathbb{E}\left[\exp\left(\lambda \cdot \sup_{\mathcal{C}} |U_n(\mathcal{C}) - u(\mathcal{C})|\right)\right] \leq \mathbb{E}\left[\exp\left(\lambda \cdot \sup_{\mathcal{C}} |A_n(\mathcal{C}) - u(\mathcal{C})|\right)\right]. \qquad (18)$$

Now, using standard symmetrization and randomization tricks, one obtains that: $\forall \lambda > 0$,

$$\mathbb{E}\left[\exp\left(\lambda \cdot \sup_{\mathcal{C}} |A_n(\mathcal{C}) - u(\mathcal{C})|\right)\right] \leq \mathbb{E}\left[\exp\left(2\lambda \cdot \mathcal{A}_{K,n}\right)\right]. \qquad (19)$$

Observing that the value of $\mathcal{A}_{K,n}$ cannot change by more than $2B/n$ when one of the $(\epsilon_i, X_i, X_{i+\lfloor n/2 \rfloor})'s$ is changed, while the others are kept fixed, the standard bounded differences inequality argument applies and yields:

$$\mathbb{E}\left[\exp\left(2\lambda \cdot \mathcal{A}_{K,n}\right)\right] \leq \exp\left(2\lambda \cdot \mathbb{E}[\mathcal{A}_{K,n}] + \frac{\lambda^2 B^2}{2n}\right). \qquad (20)$$

Next, Markov's inequality with $\lambda = (t - 2\mathbb{E}[\mathcal{A}_{K,n}])/B^2$ gives: $\mathbb{P}\{\sup_{\mathcal{C}} |A_n(\mathcal{C}) - u(\mathcal{C})| > t\} \leq \exp(-n(t - 2\mathbb{E}[\mathcal{A}_{K,n}])^2/(2B^2))$. The desired result is then immediate. $\square$

The rate bound is finally established by combining bounds (16) and (17).

**Proof of Theorem 2 (Sketch of)**

The theorem can be proved by using the decomposition (10), applying the argument above in order to control $\sup_{\mathcal{P}} |L_n(\mathcal{P})|$ and the lemma below to handle the degenerate part. The latter is based on a recent moment inequality for degenerate $U$-processes, proved in [CLV08]. Due to space limitations, technical details are left to the reader.

**Lemma 2** *(see Theorem 11 in [CLV08]) Suppose that Theorem 2's assumptions are fulfilled. There exists a universal constant $C < \infty$ such that for all $\delta \in (0,1)$, we have with probability at least $1 - \delta$: $\forall n \geq 2$,*

$$\sup_{\mathcal{P} \in \Pi_K} |M_n(\mathcal{P})| \leq K\kappa(n,\delta).$$

**Proof of Theorem 3**

The proof mimics the argument of Theorem 8.1 in [BBL05]. We thus obtain that: $\forall K \geq 1$,

$$\mathbb{E}\left[W(\widehat{\mathcal{P}}_{\widehat{K},n})\right] - W^* \leq \mathbb{E}\left[W(\widehat{\mathcal{P}}_{K,n})\right] - W^* + pen(K,n)$$
$$+ \sum_{k \geq 1} \mathbb{E}\left[\left(\sup_{\mathcal{P} \in \Pi_k} \{W(\mathcal{P}) - \widehat{W}_n(\mathcal{P})\} - pen(n,k)\right)_+\right].$$

Reproducing the argument of Theorem 1's proof, one may easily show that: $\forall k \geq 1$,

$$\mathbb{E}\left[\sup_{\mathcal{P} \in \Pi_k} \{W(\mathcal{P}) - \widehat{W}_n(\mathcal{P})\}\right] \leq 2k\mathbb{E}[\mathcal{A}_{k,n}].$$

Thus, for all $k \geq 1$, the quantity $\mathbb{P}\{\sup_{\mathcal{P} \in \Pi_k} \{W(\mathcal{P}) - \widehat{W}_n(\mathcal{P})\} \geq pen(n,k) + 2\delta\}$ is bounded by

$$\mathbb{P}\left\{\sup_{\mathcal{P} \in \Pi_k} \{W(\mathcal{P}) - \widehat{W}_n(\mathcal{P})\} \geq \mathbb{E}\left[\sup_{\mathcal{P} \in \Pi_k} \{W(\mathcal{P}) - \widehat{W}_n(\mathcal{P})\}\right] + \sqrt{(2B\log k)/n} + \delta\right\}$$
$$+ \mathbb{P}\left\{3k\mathbb{E}_\epsilon[\mathcal{A}_{k,n}] \leq 2k\mathbb{E}[\mathcal{A}_{k,n}] - \frac{27Bk\log k}{n} - \delta\right\}.$$

By virtue of the bounded differences inequality (jumps being bounded by $2B/n$), the first term is bounded by $\exp(-n\delta^2/(2B^2))/k^2$, while the second term is bounded by, $\exp(-n\delta/(9Bk))/k^3$ as shown by Lemma 8.2 in [BBL05] (see the third inequality therein). Integrating over $\delta$, one obtains:

$$\mathbb{E}\left[\left(\sup_{\mathcal{P} \in \Pi_k} \{W(\mathcal{P}) - \widehat{W}_n(\mathcal{P})\} - pen(n,k)\right)_+\right] \leq (2B\sqrt{2/n} + 18B/n)/k^2.$$

Summing next the bounds thus obtained over $k$ leads to the oracle inequality stated in the theorem.

# References

[BB06]    G. Biau and L. Bleakley. Statistical Inference on Graphs. *Statistics & Decisions*, 24:209–232, 2006.

[BBL05]   S. Boucheron, O. Bousquet, and G. Lugosi. Theory of Classification: A Survey of Some Recent Advances. *ESAIM: Probability and Statistics*, 9:323–375, 2005.

[BD04]    S. Ben-David. A framework for statistical clustering with a constant time approximation algorithms for k-median clustering. In *Proceedings of COLT'04, Lecture Notes in Computer Science, Volume 3120/2004, 415-426*, 2004.

[BDL08]   G. Biau, L. Devroye, and G. Lugosi. On the Performance of Clustering in Hilbert Space. *IEEE Trans. Inform. Theory*, 54(2):781–790, 2008.

[BvL09]   S. Bubeck and U. von Luxburg. Nearest neighbor clustering: A baseline method for consistent clustering with arbitrary objective functions. *Journal of Machine Learning Research*, 10:657–698, 2009.

[CFZ09]   B. Clarke, E. Fokoué, and H.. Zhang. *Principles and Theory for Data-Mining and Machine-Learning*. Springer, 2009.

[CLV08]   S. Clémençon, G. Lugosi, and N. Vayatis. Ranking and empirical risk minimization of U-statistics. *The Annals of Statistics*, 36(2):844–874, 2008.

[DGL96]   L. Devroye, L. Györfi, and G. Lugosi. *A Probabilistic Theory of Pattern Recognition*. Springer, 1996.

[dlPG99]  V. de la Pena and E. Giné. *Decoupling: from Dependence to Independence*. Springer, 1999.

[Dud99]   R.M. Dudley. *Uniform Central Limit Theorems*. Cambridge University Press, 1999.

[Har78]   J.A. Hartigan. Asymptotic distributions for clustering criteria. *The Annals of Statistics*, 6:117–131, 1978.

[Hoe48]   W. Hoeffding. A class of statistics with asymptotically normal distribution. *Ann. Math. Stat.*, 19:293–325, 1948.

[HTF09]   T. Hastie, R. Tibshirani, and J. Friedman. *The Elements of Statistical Learning (2nd ed.)*, pages 520–528. Springer, 2009.

[KN02]    S. Kutin and P. Niyogi. Almost-everywhere algorithmic stability and generalization error. In *Proceedings of the of the 18th Conference in Uncertainty in Artificial Intelligence*, 2002.

[Kol06]   V. Koltchinskii. Local Rademacher complexities and oracle inequalities in risk minimization (with discussion). *The Annals of Statistics*, 34:2593–2706, 2006.

[PFvN89]  R. Peck, L. Fisher, and J. van Ness. Bootstrap confidence intervals for the number of clusters in cluster analysis. *J. Am. Stat. Assoc.*, 84:184–191, 1989.

[Pol81]   D. Pollard. Strong consistency of $k$-means clustering. *The Annals of Statistics*, 9:135–140, 1981.

[Pol82]   D. Pollard. A central limit theorem for $k$-means clustering. *The Annals of Probability*, 10:919–926, 1982.

[Ser80]   R.J. Serfling. *Approximation theorems of mathematical statistics*. Wiley, 1980.

[ST08]    O. Shamir and N. Tishby. Model selection and stability in k-means clustering. In *in Proceedings of the 21rst Annual Conference on Learning Theory*, 2008.

[ST09]    O. Shamir and N. Tishby. On the reliability of clustering stability in the large sample regime. In *Advances in Neural Information Processing Systems 21*, 2009.

[TWH01]   R. Tibshirani, G. Walther, and T. Hastie. Estimating the number of clusters in a data set via the gap statistic. *J. Royal Stat. Soc.*, 63(2):411–423, 2001.

[vdV98]   A. van der Vaart. *Asymptotic Statistics*. Cambridge University Press, 1998.

[vL09]    U. von Luxburg. Clustering stability: An overview. *Foundations and Trends in Machine Learning*, 2(3):235–274, 2009.

[vLBD05]  U. von Luxburg and S. Ben-David. Towards a statistical theory of clustering. In *Pascal workshop on Statistics and Optimization of Clustering*, 2005.

[vLBD06]  U. von Luxburg and S. Ben-David. A sober look at clustering stability. In *Proceedings of the 19th Conference on Learning Theory*, 2006.

[vLBD08]  U. von Luxburg and S. Ben-David. Relating clustering stability to properties of cluster boundaries. In *Proceedings of the 21th Conference on Learning Theory*, 2008.

[WT10]    D. M. Witten and R. Tibshirani. A framework for feature selection in clustering. *J. Amer. Stat. Assoc.*, 105(490):713–726, 2010.

